# Approximate Inference and Protein-Folding

**Chen Yanover and Yair Weiss**
School of Computer Science and Engineering
The Hebrew University of Jerusalem
91904 Jerusalem, Israel
{*cheny,yweiss*}*@cs.huji.ac.il*

## Abstract

Side-chain prediction is an important subtask in the protein-folding problem. We show that finding a minimal energy side-chain configuration is equivalent to performing inference in an undirected graphical model. The graphical model is relatively sparse yet has many cycles. We used this equivalence to assess the performance of approximate inference algorithms in a real-world setting. Specifically we compared belief propagation (BP), generalized BP (GBP) and naive mean field (MF).

In cases where exact inference was possible, max-product BP always found the global minimum of the energy (except in few cases where it failed to converge), while other approximation algorithms of similar complexity did not. In the full protein data set, max-product BP always found a lower energy configuration than the other algorithms, including a widely used protein-folding software (SCWRL).

## 1 Introduction

Inference in graphical models scales exponentially with the number of variables. Since many real-world applications involve hundreds of variables, it has been impossible to utilize the powerful mechanism of probabilistic inference in these applications. Despite the significant progress achieved in approximate inference, some practical questions still remain open — it is not yet known which algorithm to use for a given problem nor is it understood what are the advantages and disadvantages of each technique. We address these questions in the context of real-world protein-folding application — the side-chain prediction problem.

Predicting side-chain conformation given the backbone structure is a central problem in protein-folding and molecular design. It arises both in ab-initio protein-folding (which can be divided into two sequential tasks — the generation of native-like backbone folds and the positioning of the side-chains upon these backbones [6]) and in homology modeling schemes (where the backbone and some side-chains are assumed to be conserved among the homologs but the configuration of the rest of the side-chains needs to be found).

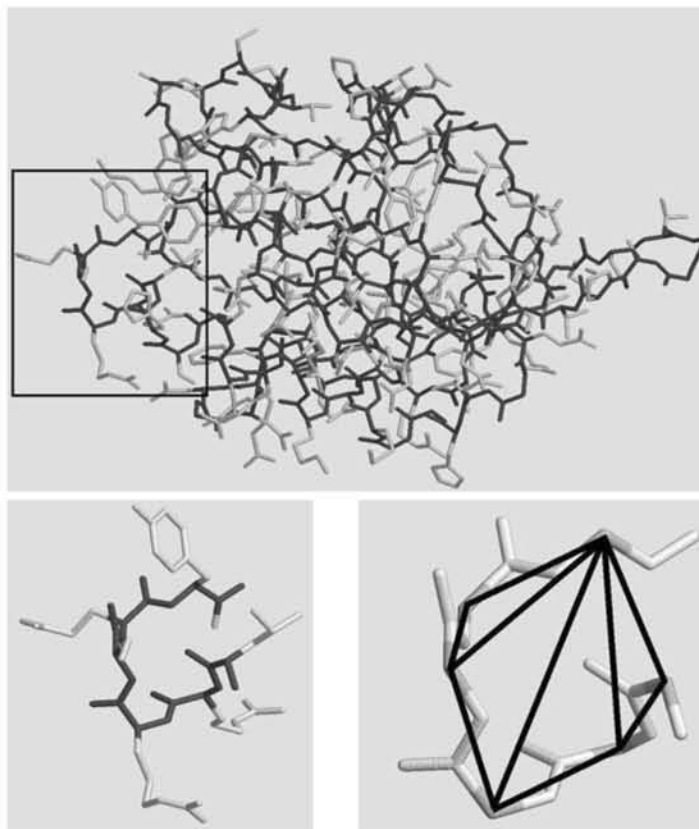

Figure 1: Cow actin binding protein (PDB code 1pne, top) and closer view of its 6 carboxyl-terminal residues (bottom-left). Given the protein backbone (black) and amino acid sequence, native side-chain conformation (gray) is searched for. Problem representation as a graphical model for those carboxyl-terminal residues shown in the bottom-right figure (nodes located at $C^\alpha$ atom positions, edges drawn in black).

In this paper, we show the equivalence between side-chain prediction and inference in an undirected graphical model. We compare the performance of BP, generalized BP and naive mean field on this problem as well as comparing to a widely used protein-folding program called SCWRL.

## 2  The side-chain prediction problem

Proteins are chains of simpler molecules called *amino acids*. All amino acids have a common structure — a central carbon atom ($C^\alpha$) to which a hydrogen atom, an amino group ($NH_2$) and a carboxyl group ($COOH$) are bonded. In addition, each amino acid has a chemical group called the *side-chain*, bound to $C^\alpha$. This group distinguishes one amino acid from another and gives its distinctive properties. Amino acids are joined end to end during protein synthesis by the formation of peptide bonds. An amino acid unit in a protein is called a *residue*. The formation of a succession of peptide bonds generate the *backbone* (consisting of $C^\alpha$ and its adjacent atoms, $N$ and $CO$, of each reside), upon which the side-chains are hanged (Figure 1).

We seek to predict the configuration of all the side-chains relative to the backbone. The standard approach to this problem is to define an energy function and use the configuration that achieves the global minimum of the energy as the prediction.

## 2.1 The energy function

We adopted the van der Waals energy function, used by SCWRL [3], which approximates the repulsive portion of Lennard-Jones 12-6 potential. For a pair of atoms, $a$ and $b$, the energy of interaction is given by:

$$E(a,b) = \begin{cases} 0 & : & d > R_0 \\ -k_2 \frac{d}{R_0} + k_2 & : & R_0 \geq d \geq k_1 R_0 \\ E_{max} & : & k_1 R_0 > d \end{cases} \qquad (1)$$

where $E_{max} = 10, k_1 = 0.8254$ and $k_2 = \frac{E_{max}}{1-k_1}$. $d$ denotes the distance between $a$ and $b$ and $R_0$ is the sum of their radii. Constant radii were used for protein's atoms (Carbon - 1.6Å, Nitrogen and Oxygen - 1.3Å, Sulfur - 1.7Å). For two sets of atoms, the interaction energy is a sum of the pairwise atom interactions. The energy surface of a typical protein in the data set has dozens to thousands local minima.

## 2.2 Rotamers

The configuration of a single side-chain is represented by at most 4 dihedral angles (denoted $\chi_1, \chi_2, \chi_3$ and $\chi_4$). Any assignment of $\chi$ angles for all the residues defines a protein configuration. Thus the energy minimization problem is a highly nonlinear continuous optimization problem.

It turns out, however, that side-chains have a small repertoire of energetically preferred conformations, called *rotamers*. Statistical analysis of those conformations in well-determined protein structures produce a rotamer library. We used a backbone dependent rotamer library (by Dunbrack and Kurplus, July 2001 version). Given the coordinates of the backbone atoms, its dihedral angles $\phi$ (defined, for the $i^{th}$ residue, by $C_{i-1} - N_i - C_i^\alpha - C_i$) and $\psi$ (defined by $N_i - C_i^\alpha - C_i - N_{i+1}$) were calculated. The library then gives the typical rotamers for each side-chain and their prior probabilities.

By using the library we convert the continuous optimization problem into a discrete one. The number of discrete variables is equal to the number of residues and the possible values each variable can take lies between 2 and 81.

## 2.3 Graphical model

Since we have a discrete optimization problem and the energy function is a sum of pairwise interactions, we can transform the problem into a graphical model with pairwise potentials. Each node corresponds to a residue, and the state of each node represents the configuration of the side chain of that residue. Denoting by $\{r_i\}$ an assignment of rotamers for all the residues then:

$$P(\{r_i\}) = \frac{1}{Z} e^{-\frac{1}{T} E(\{r_i\})} = \frac{1}{Z} e^{-\frac{1}{T} \sum_{ij} E(r_i) + E(r_i, r_j)}$$

$$= \frac{1}{Z} \prod_i \Psi_i(r_i) \prod_{i,j} \Psi_{ij}(r_i, r_j) \qquad (2)$$

where $Z$ is an explicit normalization factor and $T$ is the system "temperature" (used as free parameter). The local potential $\Psi_i(r_i)$ takes into account the prior

probability of the rotamer $\tilde{p}_i(r_i)$ (taken from the rotamer library) and the energy of the interactions between that rotamer and the backbone:

$$\Psi_i(r_i) = \tilde{p}_i(r_i) e^{-\frac{1}{T}E(r_i,backbone)} \qquad (3)$$

Equation 2 requires multiplying $\Psi_{ij}$ for all pairs of residues $i, j$ but note that equation 1 gives zero energy for atoms that are sufficiently far away. Thus we only need to calculate the pairwise interactions for nearby residues. To define the topology of the undirected graph, we examine all pairs of residues $i, j$ and check whether there exists an assignment $r_i, r_j$ for which the energy is nonzero. If it exists, we connect nodes $i$ and $j$ in the graph and set the potential to be:

$$\Psi_{ij}(r_i, r_j) = e^{-\frac{1}{T}E(r_i,r_j)} \qquad (4)$$

Figure 1 shows a subgraph of the undirected graph. The graph is relatively sparse (each node is connected to nodes that are close in 3D space) but contains many small loops. A typical protein in the data set gives rise to a model with hundreds of loops of size 3.

## 3 Experiments

When the protein was small enough we used the max-junction tree algorithm [1] to find the most likely configuration of the variables (and hence the global minimum of the energy function). Murphy's implementation of the JT algorithm in his BN toolbox for Matlab was used [10].

The approximate inference algorithms we tested were loopy belief propagation (BP), generalized BP (GBP) and naive mean field (MF).

BP is an exact and efficient local message passing algorithm for inference in singly connected graphs [15]. Its essential idea is replacing the exponential enumeration (either summation or maximizing) over the unobserved nodes with series of local enumerations (a process called "elimination" or "peeling"). Loopy BP, that is applying BP to multiply connected graphical models, may not converge due to circulation of messages through the loops [12]. However, many groups have recently reported excellent results using loopy BP as an approximate inference algorithm [4, 11, 5]. We used an asynchronous update schedule and ran for 50 iterations or until numerical convergence.

GBP is a class of approximate inference algorithms that trade complexity for accuracy [15]. A subset of GBP algorithms is equivalent to forming a graph from clusters of nodes and edges in the original graph and then running ordinary BP on the cluster graph. We used two large clusters. Both clusters contained all nodes in the graph but each cluster contained only a subset of the edges. The first cluster contained all edges resulting from residues, for which the difference between its indices is less than a constant $k$ (typically, 6). All other edges were included in the second cluster. It can be shown that the cluster graph BP messages can be computed efficiently using the JT algorithm. Thus this approximation tries to capture dependencies between a large number of nodes in the original graph while maintaining computational feasibility.

The naive MF approximation tries to approximate the joint distribution in equation 2 as a product of independent marginals $q_i(r_i)$. The marginals $q_i(r_i)$ can be found by iterating:

$$q_i(r_i) \leftarrow \alpha \Psi_i(r_i) \exp\left( \sum_{j \in N_i} \sum_{r_j} q_j(r_j) \log \Psi_{ij}(r_i, r_j) \right) \qquad (5)$$

where $\alpha$ denotes a normalization constant and $N_i$ means all nodes neighboring $i$. We initialized $q_i(r_i)$ to $\Psi_i(r_i)$ and chose a random update ordering for the nodes. For each protein we repeated this minimization 10 times (each time with a different update order) and chose the local minimum that gave the lowest energy.

In addition to the approximate inference algorithms described above, we also compared the results to two approaches in use in side-chain prediction: the SCWRL and DEE algorithms. The Side-Chain placement With a Rotamer Library (SCWRL) algorithm is considered one of the leading algorithms for predicting side-chain conformations [3]. It uses the energy function described above (equation 1) and a heuristic search strategy to find a minimal energy conformation in a discrete conformational space (defined using rotamer library).

Dead end elimination (DEE) is a search algorithm that tries to reduce the search space until it becomes suitable for an exhaustive search. It is based on a simple condition that identifies rotamers that cannot be members of the global minimum energy conformation [2]. If enough rotamers can be eliminated, the *global minimum energy conformation* can be found by an exhaustive search of the remaining rotamers.

The various inference algorithms were tested on set of 325 X-ray crystal structures with resolution better than or equal to 2Å, R factor below 20% and length up to 300 residues. One representative structure was selected from each cluster of homologous structures (50% homology or more). Protein structures were acquired from Protein Data Bank site (http://www.rcsb.org/pdb).

Many proteins contain Cysteine residues which tend to form strong disulfide bonds with each other. A standard technique in side-chain prediction (used e.g. in SCWRL) is to first search for possible disulfide bonds and if they exist to freeze these residues in that configuration. This essentially reduces the search space. We repeated our experiments with and without freezing the Cysteine residues.

Side-chain to backbone interaction seems to be much severe than side-chain to side-chain interaction — the backbone is more rigid than side-chains and its structure assumed to be known. Therefore, the parameter $R$ was introduced into the pairwise potential equation, as follows:

$$\Psi_{ij}(r_i, r_j) = (e^{-\frac{1}{T}E(r_i, r_j)})^{\frac{1}{R}} \tag{6}$$

Using $R > 1$ assigns an increased weight for side-chain to backbone interactions over side-chain to side-chain interactions. We repeated our experiments both with $R = 1$ and $R > 1$. It worth mentioning that SCWRL implicitly adopts a weighting assumption that assigns an increased weight to side-chain to backbone interactions.

## 4   Results

In our first set of experiments we wanted to compare approximate inference to exact inference. In order to make exact inference possible we restricted the possible rotamers of each residue. Out of the 81 possible states we chose a subset whose local probability accounted for 90% of the local probability. We constrained the size of the subset to be at least 2. The resulting graphical model retains only a small fraction of the loops occurring in the full graphical model (about 7% of the loops of size 3). However, it still contains many small loops, and in particular, dozens of loops of size 3.

On these graphs we found that ordinary max-product BP *always found the global minimum* of the energy function (except in few cases where it failed to converge).

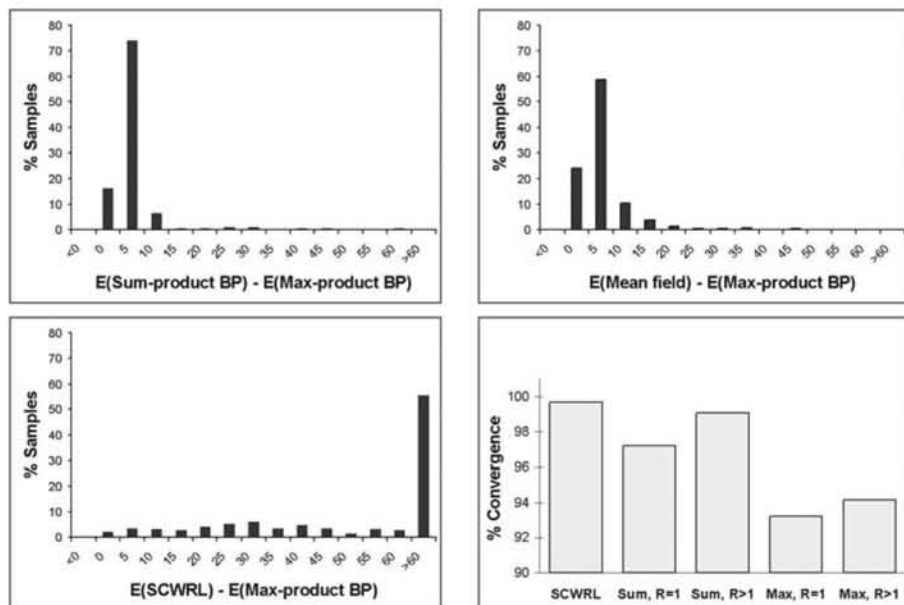

Figure 2: Sum-product BP (top-left), naive MF (top-right) and SCWRL (bottom-left) algorithms energies are always higher than or equal to max-product BP energy. Convergence rates for the various algorithms shown in bottom-right chart.

Sum-product BP failed to find sum-JT conformation in 1% of the graphs only. In contrast the naive MF algorithm found the global minimum conformation for 38% of the proteins and on 17% of the runs only. The GBP algorithm gave the same result as the ordinary BP but it converged more often (e.g. 99.6% and 98.9% for sum-product GBP and BP, respectively).

In the second set of experiments we used the full graphical models. Since exact inference is impossible we can only compare the relative energies found by the different approximate inference algorithms. Results are shown in Figure 2. Note that, when it converged, max-product BP *always found a lower energy configuration compared to the other algorithms*. This finding agrees with the observation that the max-product solution is a "neighborhood optimum" and therefore guaranteed to be better than all other assignments in a large region around it [13].

We also tried decreasing $T$, the system "temperature", for sum-product (in the limit of zero temperature it should approach max-product). In 96% of the time, using lower temperature ($T = 0.3$ instead of $T = 1$) indeed gave a lower energy configuration. Even at this reduced temperature, however, max-product always found a lower energy configuration.

All algorithms converged in more than 90% of the cases. However, sum-product converged more often than max-product (Figure 2, bottom-right). Decreasing temperature resulted in lower convergence rate for sum-product BP algorithm (e.g. 95.7% compared to 98.15% in full size graphs using disulfide bonds). It should be mentioned that SCWRL failed to converge on a single protein in the data set.

Applying the DEE algorithm to the side-chain prediction graphical models dramatically decreased the size of the conformational search space, though, in most cases, the resulted space was still infeasible. Moreover, max-product BP was indifferent

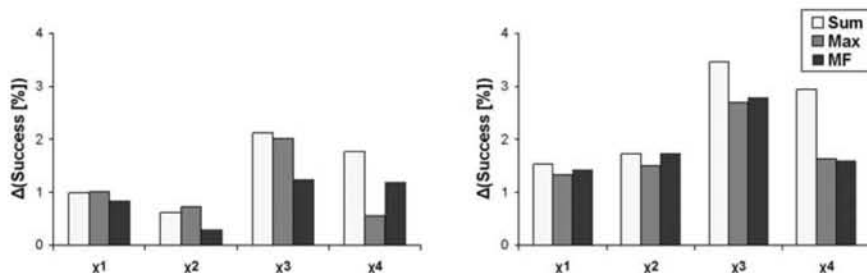

| SCWRL buried residues success rates | | | |
|---|---|---|---|
| $\chi_1$ | $\chi_2$ | $\chi_3$ | $\chi_4$ |
| 85.9% | 62.2% | 40.3% | 25.5% |

Figure 3: Inference results - success rate. SCWRL buried residues success rate subtracted from sum-product BP (light gray), max-product BP (dark gray) and MF (black) rates when equally weighting side-chain to backbone and side-chain to side-chain clashes (left) and assigning increased weight for side-chain to backbone clashes (right).

to that space reduction — it failed to converge for the same models and, when converged, found the same conformation.

## 4.1 Success rate

In comparing the performance of the algorithms, we have focused on the energy of the found configuration since this is the quantity the algorithms seek to optimize. A more realistic performance measure is: how well do the algorithms predict the native structure of the protein?

The dihedral angle $\chi_i$ is deemed correct when it is within $40°$ of the native (crystal) structure and $\chi_1$ to $\chi_{i-1}$ are correct. Success rate is defined as the portion of correctly predicted dihedral angles.

The success rates of the conformations, inferred by both max- and sum-product outperformed SCWRL's (Figure 3). For buried residues (residues with relative accessibility lower than 30% [9]) both algorithms added 1% to SCWRL's $\chi_1$ success rate. Increasing the weight of side-chain to backbone interactions over side-chain to side-chain interactions resulted in better success rates (Figure 3, right). Freezing Cysteine residues to allow the formation of disulfide bonds slightly increased the success rate.

## 5   Discussion

Recent years have shown much progress in approximate inference. We believe that the comparison of different approximate inference algorithms is best done in the context of a real-world problem. In this paper we have shown that for a real-world problem with many loops, the performance of belief propagation is excellent. In problems where exact inference was possible max-product BP always found the global minimum of the energy function and in the full protein data set, max-product BP always found a lower energy configuration compared to the other algorithms tested.

SCWRL is considered one of the leading algorithms for modeling side-chain conformations. However, in the last couple of years several groups reported better results due to more accurate energy function [7], better searching algorithm [8], or extended rotamer library [14].

As shown, by using inference algorithms we achieved low energy conformations, compared to existing algorithms. However, this leads only to a modest increase in prediction accuracy. Using an energy function, which gives a better approximation to the "true" physical energy (and particularly, assigns lowest energy to the native structure) should significantly improve the success rate. A promising direction for future research is to try and learn the energy function from examples. Inference algorithms such as BP may play an important role in the learning procedure.

# References

[1] R. Cowell. Introduction to inference in Bayesian networks. In Michael I. Jordan, editor, *Learning in Graphical Models*. Morgan Kauffmann, 1998.

[2] Johan Desmet, Marc De Maeyer, Bart Hazes, and Ignace Lasters. The dead-end elmination theorem and its use in protein side-chain positioning. *Nature*, 356:539–542, 1992.

[3] Roland L. Dunbrack, Jr. and Martin Kurplus. Back-bone dependent rotamer library for proteins: Application to side-chain predicrtion. *J. Mol. Biol*, 230:543–574, 1993. See also http://www.fccc.edu/research/labs/dunbrack/scwrl/.

[4] William T. Freeman and Egon C. Pasztor. Learning to estimate scenes from images. In M.S. Kearns, S.A. Solla, and D.A. Cohn, editors, *Adv. Neural Information Processing Systems 11*. MIT Press, 1999.

[5] Brendan J. Frey, Ralf Koetter, and Nemanja Petrovic. Very loopy belief propagation for unwrapping phase images. In *Adv. Neural Information Processing Systems 14*. MIT Press, 2001.

[6] Enoch S. Huang, Patrice Koehl, Michael Levitt, Rohit V. Pappu, and Jay W. Ponder. Accuracy of side-chain prediction upon near-native protein backbones generated by ab initio folding methods. *Proteins*, 33(2):204–217, 1998.

[7] Shide Liang and Nick V. Grishin. Side-chain modeling with an optimized scoring function. *Protein Sci*, 11(2):322–331, 2002.

[8] Loren L. Looger and Homme W. Hellinga. Generalized dead-end elimination algorithms make large-scale protein side-chain structure prediction tractable: implications for protein design and structural genomics. *J Mol Biol*, 307(1):429–445, 2001.

[9] Joaquim Mendes, Cludio M. Soare, and Maria Armnia Carrondo. mprovement of side-chain modeling in proteins with the self-consistent mean field theory method based on an analysis of the factors influencing prediction. *Biopolymers*, 50(2):111–131, 1999.

[10] Kevin Murphy. The bayes net toolbox for matlab. *Computing Science and Statistics*, 33, 2001.

[11] Kevin P. Murphy, Yair Weiss, and Micheal I. Jordan. Loopy belief propagation for approximate inference: an empirical study. In *Proceedings of Uncertainty in AI*, 1999.

[12] Judea Pearl. *Probabilistic Reasoning in Intelligent Systems: Networks of Plausible Inference*. Morgan Kaufmann, 1988.

[13] Yair Weiss and William T. Freeman. On the optimality of solutions of the max-product belief propagation algorithm. *IEEE Transactions on Information Theory*, 47(2):723–735, 2000.

[14] Zhexin Xiang and Barry Honig. Extending the accuracy limits of prediction for side-chain conformations. *J Mol Biol*, 311(2):421–430, 2001.

[15] Jonathan S. Yedidia, William T. Freeman, and Yair Weiss. Understanding belief propagation and its generalization. In G. Lakemayer and B. Nebel, editors, *Exploring Artificial Intelligence in the New Millennium*. Morgan Kauffmann, 2002.
